# A Connectionist Learning Control Architecture for Navigation

Jonathan R. Bachrach
Department of Computer and Information Science
University of Massachusetts
Amherst, MA 01003

## Abstract

A novel learning control architecture is used for navigation. A sophisticated test-bed is used to simulate a cylindrical robot with a sonar belt in a planar environment. The task is short-range homing in the presence of obstacles. The robot receives no global information and assumes no comprehensive world model. Instead the robot receives only sensory information which is inherently limited. A connectionist architecture is presented which incorporates a large amount of a priori knowledge in the form of hard-wired networks, architectural constraints, and initial weights. Instead of hard-wiring static potential fields from object models, my architecture learns sensor-based potential fields, automatically adjusting them to avoid local minima and to produce efficient homing trajectories. It does this without object models using only sensory information. This research demonstrates the use of a large modular architecture on a difficult task.

## 1 OVERVIEW

I present a connectionist learning control architecture tailored for simulated short-range homing in the presence of obstacles. The kinematics of a cylindrical robot (shown in Figure 1) moving in a planar environment is simulated. The robot has wheels that propel it independently and simultaneously in both the x and y directions with respect to a fixed orientation. It can move up to one radius per discrete time step. The robot has a 360 degree sensor belt with 16 distance sensors and 16 grey-scale sensors evenly placed around its perimeter. These 32 values form the robot's *view*.

Figure 2 is a display created by the navigation simulator. The bottom portion of

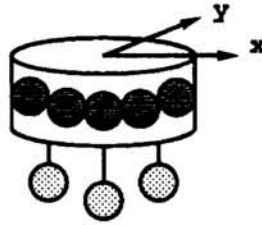

Figure 1: Simulated robot.

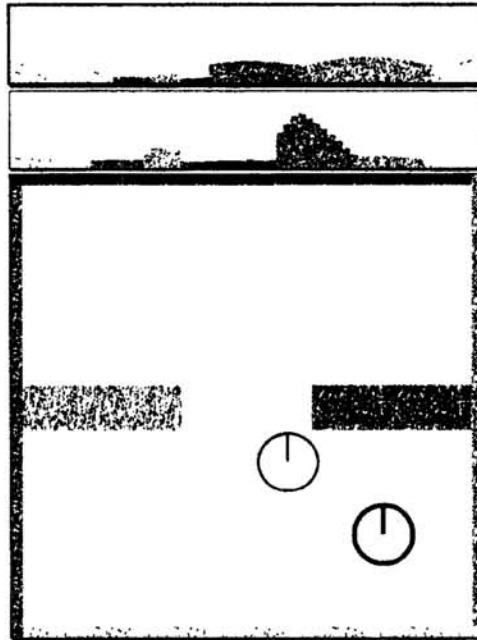

Figure 2: Navigation simulator.

the figure shows a bird's-eye view of the robot's environment. In this display, the bold circle represents the robot's "home" position, with the radius line indicating the home orientation. The other circle with radius line represents the robot's current position and orientation. The top panel shows the grey-scale view from the home position, and the next panel down shows the grey-scale view from the robot's current position. For better viewing, the distance and grey-scale sensor values are superimposed, and the height of the profile is 1/distance instead of distance. Thus as the robot gets closer to objects they get taller, and when the robot gets farther away from objects they get shorter in the display.

The robot cannot move through nor "see" through obstacles (i.e., obstacles are opaque). The task is for the robot to align itself with the home position from arbitrary starting positions in the environment while not colliding with obstacles. This task is performed using only the sensory information—the robot does not have access to the bird's-eye view.

This is a difficult control task. The sensory information forms a high-dimensional

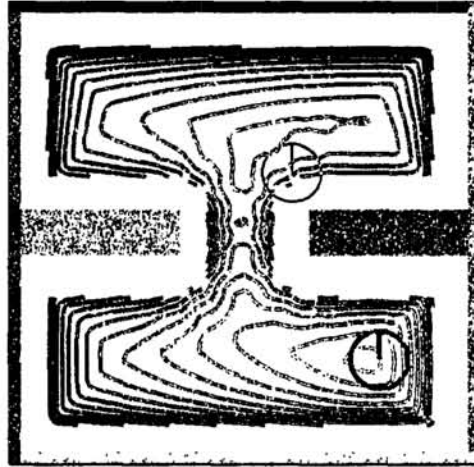

Figure 3: The potential field method. This figure shows a contour plot of a terrain created using potential fields generated from object models. The contour diagram shows level curves where the grey level of the line depicts the height of the line: the maximum height is depicted in black, and the minimum height is depicted in white.

continuous space, and successful homing generally requires a nonlinear mapping from this space to the space of real-valued actions. Further, training networks is not easily achieved on this space. The robot assumes no comprehensive world model and receives no global information, but receives only sensory information that is inherently limited. Furthermore, it is difficult to reach home using random exploration thereby making simple trial-and-error learning intractable. In order to handle this task an architecture was designed that facilitates the coding of domain knowledge in the form of hard-wired networks, architectural constraints, and initial weights.

## 1.1   POTENTIAL FIELDS

Before I describe the architecture, I briefly discuss a more traditional technique for navigation that uses potential fields. This technique involves building explicit object models representing the extent and position of objects in the robot's environment. Repelling potential fields are then placed around obstacles using the object models, and an attracting potential field is placed on the goal. This can be visualized as a terrain where the global minimum is located at the goal, and where there are bumps around the obstacles. The robot goes home by descending the terrain. The contour diagram in Figure 3 shows such a terrain. The task is to go from the top room to the bottom through the door. Unfortunately, there can be local minima. In this environment there are two prime examples of minima: the right-hand wall between the home location and the upper room—opposing forces exactly counteract each other to produce a local minimum in the right-hand side of the upper room, and the doorway—the repelling fields on the door frame create an insurmountable bump in the center of the door.

In contrast, my technique learns a sensor-based potential field model. Instead of hard-wiring static potential fields from the object models, the proposed architecture

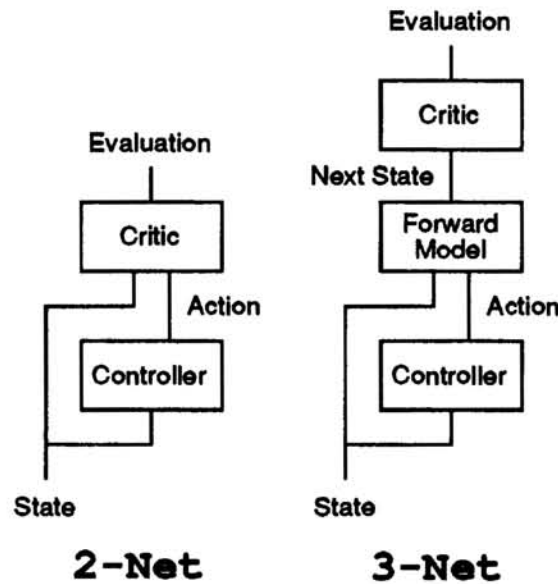

Figure 4: Control architectures.

learns potential fields, automatically adjusting them to both avoid local minima and produce efficient trajectories. Furthermore, it does this without object models, using only sensory information.

## 1.2   2-NET/3-NET ARCHITECTURES

I shall begin by introducing two existing architectures: the 2-net and 3-net architectures. These architectures were proposed by Werbos [9] and Jordan and Jacobs [4] and are also based on the ideas of Barto, Sutton, Watkins [2, 1, 8], and Jordan and Rumelhart [3]. The basic idea is to learn an evaluation function and then train the controller by differentiating this function with respect to the controller weights. These derivatives indicate how to change the controller's weights in order to minimize or maximize the evaluation function. The 2-net architecture consists of a controller and a critic. The controller maps states to actions, and the 2-net critic maps state/action pairs to evaluations. The 3-net architecture consists of a controller, a forward model, and a critic. The controller maps states to actions, the forward model maps state/action pairs to next states, and the 3-net critic maps states to evaluations.

It has been said that it is easier to train a 2-net architecture because there is no forward model [5]. The forward model might be very complicated and difficult to train. With a 2-net architecture, only a 2-net critic is trained based on state/action input pairs. But what if a forward model already exists or even a priori knowledge exists to aid in explicit coding of a forward model? Then it might be simpler to use the 3-net architecture because the 3-net critic would be easier to train. It is based on state-only input and not state/action pairs, and it includes more domain knowledge.

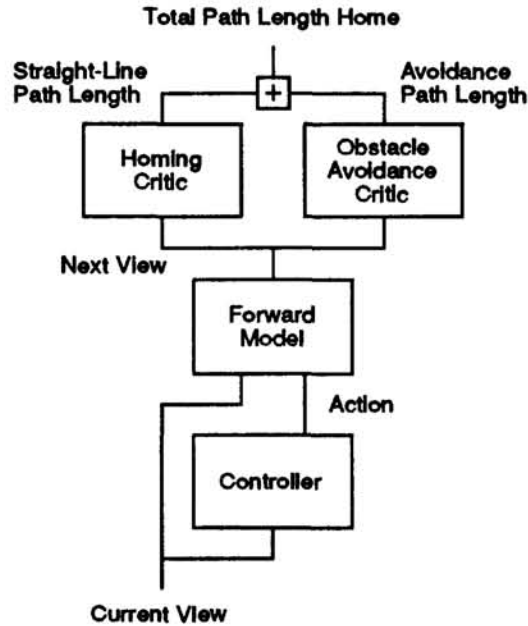

Figure 5: My architecture.

## 2   THE NAVIGATION ARCHITECTURE

The navigation architecture is a version of a 3-net architecture tailored for navigation, where the state is the robot's view and the evaluation is an estimate of the length of the shortest path for the robot's current location to home. It consists of a controller, a forward model, and two adaptive critics. The controller maps views to actions, the forward model maps view/action pairs to next views, the homing critic maps views to path length home using a straight line trajectory, and the obstacle avoidance critic maps views to additional path length needed to avoid obstacles. The sum of the outputs of the homing critic and the obstacle avoidance critic equals the total path length home. The forward model is a hard-wired differentiable network incorporating geometrical knowledge about the sensors and space. Both critics and the controller are radial basis networks using Gaussian hidden units.

### 2.1   TRAINING

Initially the controller is trained to produce straight-line trajectories home. With the forward model fixed, the homing critic and the controller are trained using dead-reckoning. Dead-reckoning is a technique for keeping track of the distance home by accumulating the incremental displacements. This distance provides a training signal for training the homing critic via supervised learning.

Next, the controller is further trained to avoid obstacles. In this phase, the obstacle avoidance critic is added while the weights of the homing critic and forward model are frozen. Using the method of temporal differences [7] the controller and obstacle avoidance critic are adjusted so that the expected path length decreases by one radius per time step. After training, the robot takes successive one-radius steps

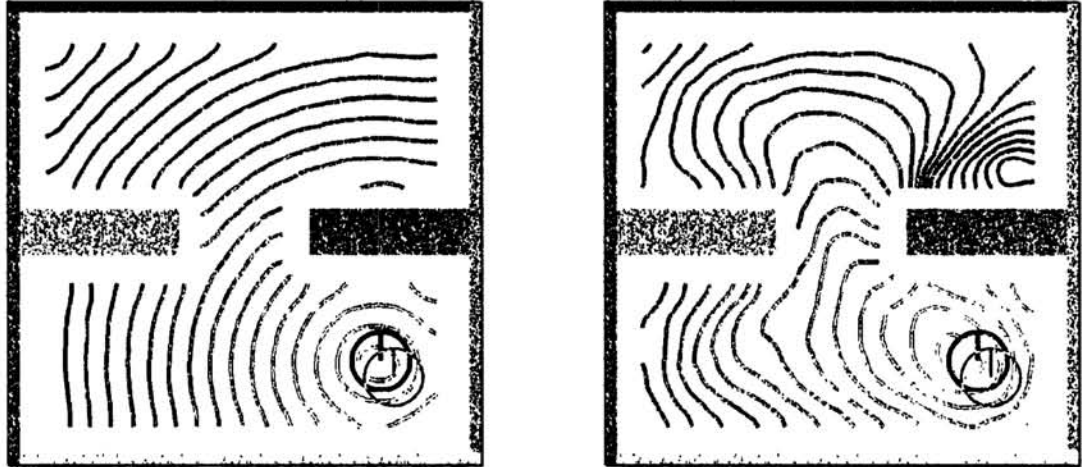

Figure 6: An example.

toward its home location.

## 3   AN EXAMPLE

I applied this architecture to the environment shown in Figure 2. Figure 6 shows the results of training. The left panel is a contour plot of the output of the homing critic and reflects only the straight-line distance to the home location. The right panel is a contour plot of the combined output of the homing critic and the obstacle avoidance critic and now reflects the actual path length home. After training the robot is able to form efficient homing trajectories starting from anywhere in the environment.

## 4   DISCUSSION

The homing task represents a difficult control task requiring the solution of a number of problems. The first problem is that there is a small chance of getting home using random exploration. The solution to this problem involves building a nominal initial controller that chooses straight-line trajectories home. Next, because the state space is high-dimensional and continuous it is impractical to evenly place Gaussian units, and it is difficult to learn continuous mappings using logistic hidden units. Instead I use Gaussian units whose initial weights are determined using expectation maximization. This is a soft form of competitive learning [6] that, in my case, creates spatially tuned units. Next, the forward model for the robot's environments is very difficult to learn. For this reason I used a hard-wired forward model whose performance is good in a wide range of environments. Here the philosophy is to learn only things that are difficult to hard-wire. Finally, the 2-net critic is difficult to train. Therefore, I split the 2-net critic into a 3-net critic and a hard-wired forward model.

There are many directions for extending this work. First, I would like to apply this architecture to real robots using realistic sensors and dynamics. Secondly, I want to

to look at long range homing. Lastly, I would like to investigate navigation tasks involving multiple goals.

**Acknowledgements**

This material is based upon work supported by the Air Force Office of Scientific Research, Bolling AFB, under Grant AFOSR-89-0526 and by the National Science Foundation under Grant ECS-8912623. I would like to thank Richard Durbin, David Rumelhart, Andy Barto, and the UMass Adaptive Networks Group for their help on this project.

# References

[1] A. G. Barto, R. S. Sutton, and C. Watkins. Sequential decision problems and neural networks. In David S. Touretzky, editor, *Advances in Neural Information Processing Systems*, P.O. Box 50490, Palo Alto, CA 94303, 1989. Morgan Kaufmann Publishers.

[2] Andrew G. Barto, Richard S. Sutton, and Charles W. Anderson. Neuronlike adaptive elements that can solve difficult learning control problems. *IEEE Transactions on Systems, Man, and Cybernetics*, SMC-13(15), September/October 1985.

[3] M. I. Jordan and D. E. Rumelhart. Supervised learning with a distal teacher. 1989. Submitted to: *Cognitive Science*.

[4] Michael I. Jordan and Robert Jacobs. Learning to control an unstable system with forward modeling. In David S. Touretzky, editor, *Advances in Neural Information Processing Systems*, P.O. Box 50490, Palo Alto, CA 94303, 1989. Morgan Kaufmann Publishers.

[5] Sridhar Mahadevan and Jonathan Connell. Automatic programming of behavior-based robots using reinforcemnt learning. Technical report, IBM Research Division, T.J. Watson Research Center, Box 704, Yorktown Heights, NY 10598, 1990.

[6] S. J. Nowlan. A generative framework for unsupervised learning. Denver, Colorado, 1989. IEEE Conference on Neural Information Processing Systems— Natural and Synthetic.

[7] Richard Sutton. Learning to predict by the methods of temporal differences. Technical report, GTE Laboratories, 1987.

[8] Richard S. Sutton. *Temporal Credit Assignment in Reinforcement Learning*. PhD thesis, Department of Computer and Information Science, University of Massachusetts at Amherst, 1984.

[9] Paul J. Werbos. Reinforcement learning over time. In T. Miller, R. S. Sutton, and P. J. Werbos, editors, *Neural Networks for Control*. The MIT Press, Cambridge, MA, In press.
